# Recognition-based Segmentation of On-line Cursive Handwriting

**Nicholas S. Flann**
Department of Computer Science
Utah State University
Logan, UT 84322-4205
flann@nick.cs.usu.edu

## Abstract

This paper introduces a new recognition-based segmentation approach to recognizing on-line cursive handwriting from a database of 10,000 English words. The original input stream of $x, y$ pen coordinates is encoded as a sequence of uniform stroke descriptions that are processed by six feed-forward neural-networks, each designed to recognize letters of different sizes. Words are then recognized by performing best-first search over the space of all possible segmentations. Results demonstrate that the method is effective at both writer dependent recognition (1.7% to 15.5% error rate) and writer independent recognition (5.2% to 31.1% error rate).

## 1 Introduction

With the advent of pen-based computers, the problem of automatically recognizing handwriting from the motions of a pen has gained much significance. Progress has been made in reading disjoint block letters [Weissman et. al, 93]. However, cursive writing is much quicker and natural for humans, but poses a significant challenge to pattern recognition systems because of its variability, ambiguity and need to both segment and recognize the individual letters. Recent techniques employing self-organizing networks are described in [Morasso et. al, 93] and [Schomaker, 1993]. This paper presents an alternative approach based on feed-forward networks.

On-line handwriting consists of writing with a pen on a touch-terminal or digitizing

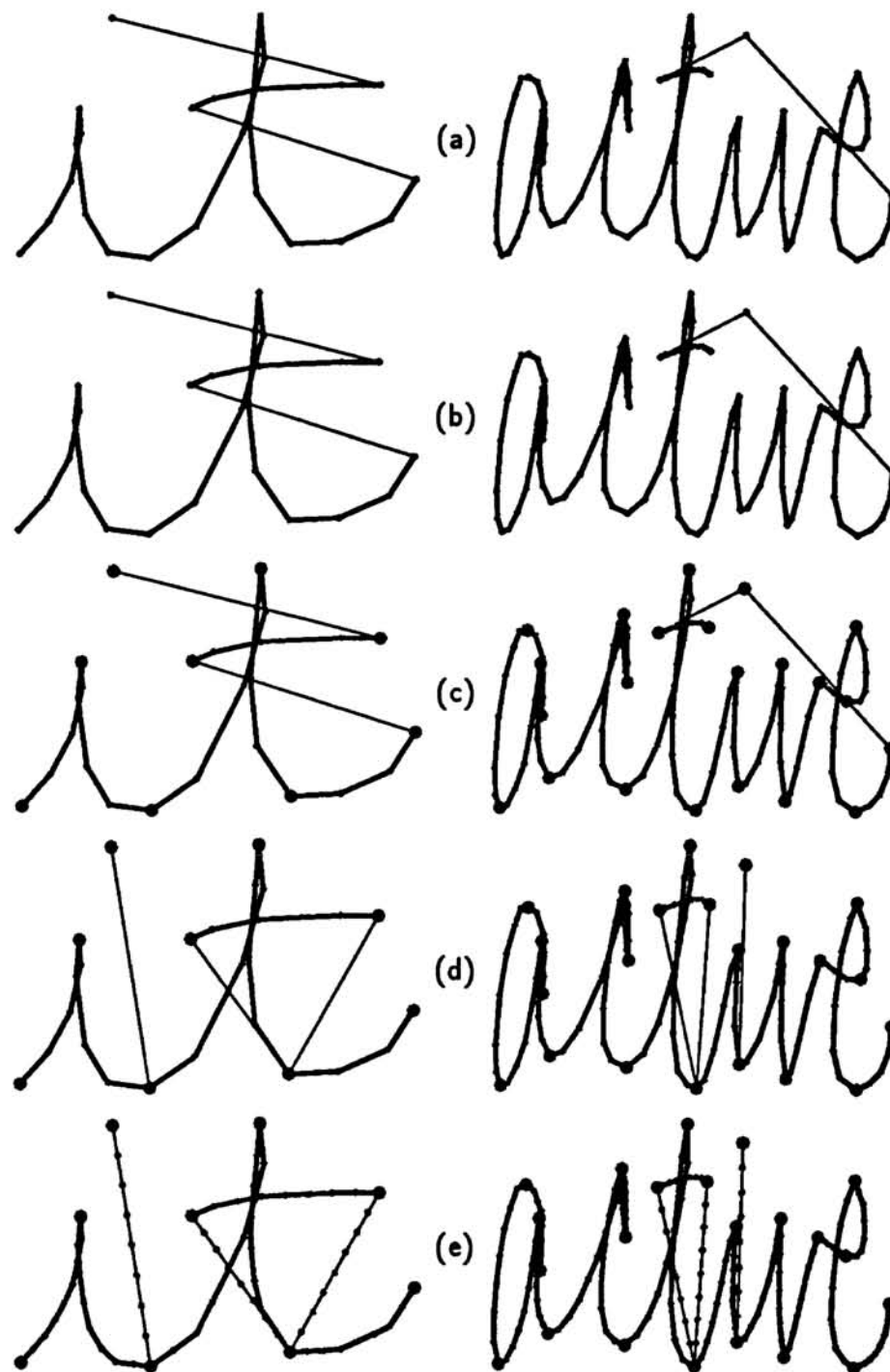

Figure 1: The five principal stages of preprocessing: (a) The original data, $x, y$ values sampled every $10mS$. (b) The slant is normalized through a shear transformation; (c) Stroke boundaries are determined at points where $y$ velocity equals 0 or pen-up or pen-down events occur; (d) Delayed strokes are reordered and associated with corresponding strokes of the same letters; (e) Each stroke is resampled in space to correspond to exactly 8 points. Note pen-down strokes are shown as thick lines, pen-up strokes as thin lines.

tablet. The device produces a regular stream of $x, y$ coordinates, describing the positions of the pen while writing. Hence the problem of recognizing on-line cursively written words is one of mapping a variable length sequence of $x, y$ coordinates to a variable length sequence of letters. Developing a system that can accurately perform this mapping faces two principal problems: First, because handwriting is done with little regularity in speed, there is unavoidable variability in input size. Second, because no pen-up events or spatial gaps signal the end of one letter and the beginning of the next, the system must perform both segmentation and recognition.

This second problem necessitates the development of a recognition-based segmentation approach. In [Schenkel *et al.*, 93] one such approach is described for connected block letter recognition where the system learns to recognize segmentation points. In this paper an alternative method is presented that first performs exhaustive recognition then searches the space of possible segmentations. The remainder of the paper describes the method in more detail and presents results that demonstrate its effectiveness at recognizing a variety of cursive handwriting styles.

## 2  Methodology

The recognition system consists of three subsystems: (a) the *preprocessor* that maps the initial stream of $x, y$ coordinates to a stream of stroke descriptions; (b) the *letter classifier* that learns to recognize individual letters of different size; and (c) the *word finder* that performs recognition-based segmentation over the output of the letter classifier to identify the most likely word written.

### 2.1  Preprocessing

The preprocessing stage follows steps outlined in [Guerfali & Plamondon, 93] and is illustrated in Figure 1. First the original data is smoothed by passing it through a low-pass filter, then reslanted to make the major stroke directions vertical. This is achieved by computing the mean angle of all the individual lines then applying a shear transformation to remove it. Second, the strokes boundaries are identified as points when $\dot{y} = 0$ or when the pen is picked up or put down. Zero $y$ velocity was chosen rather than minimum absolute velocity [Morasso *et. al*, 93] since it was found to be more robust. Third, delayed strokes such as those that dot an $i$ or cross a $t$ are reordered to be associated with their corresponding letter. Here the delayed stroke is placed to immediately follow the closest down stroke and linked into the stroke sequence by straight line pen-up strokes. Fourth, each stroke is resampled in the space domain (using linear interpolation) so as to represent it as exactly eight $x, y$ coordinates. Finally the new stream of $x, y$ coordinates is converted to a stream of 14 feature values.

Eight of these features are similar to those used in [Weissman *et. al*, 93], and represent the angular acceleration (as the sin and cos of the angle), the angular velocity of the line (as the sin and cos of the angle), the $x, y$ coordinates ($x$ has a linear ramp removed), and first differential $\delta x, \delta y$. One feature denotes whether the pen was down or up when the line was drawn. The remaining features encode more abstract information about the stroke.

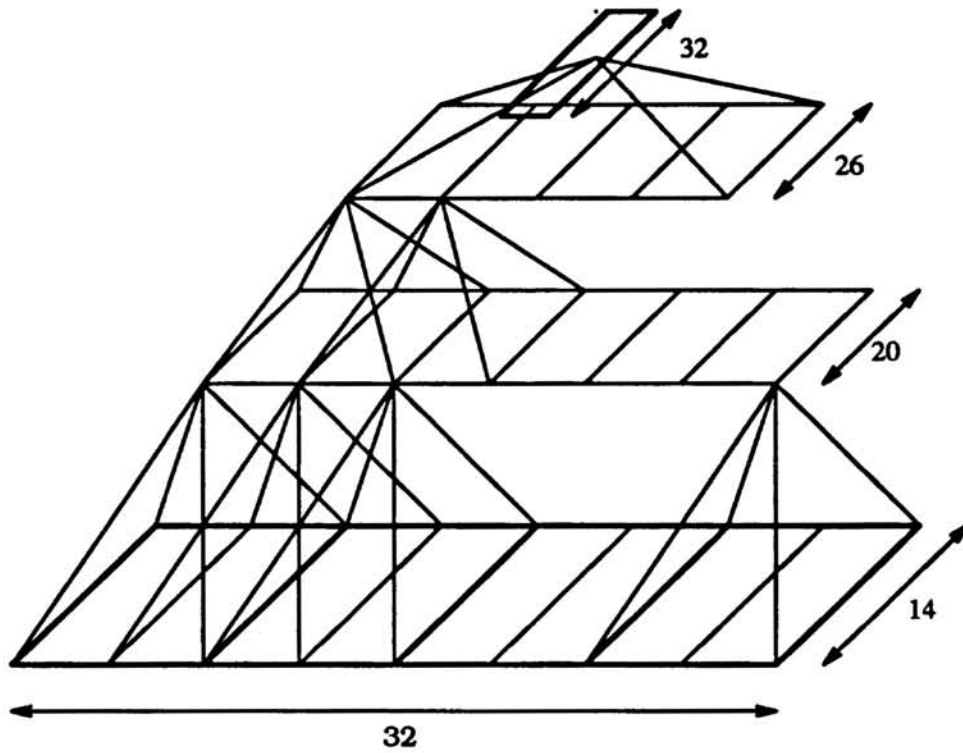

Figure 2: The pyramid-style architecture of the network used to recognize 2 stroke letters. The input size is $32 \times 14$; 32 is from the 4 input strokes (each represented by 8 resampled points), two central strokes from the letter and the 2 context strokes, one each side; 14 is from the number of features employed to represent each point. Not all the receptive fields are shown. The first hidden layer consists of 7 fields, 4 over each stroke and 3 more spanning the stroke boundaries. The next hidden layer consists of 5 fields, each spanning $3 \times 20$ inputs. The output is a 32 bit error-correcting code.

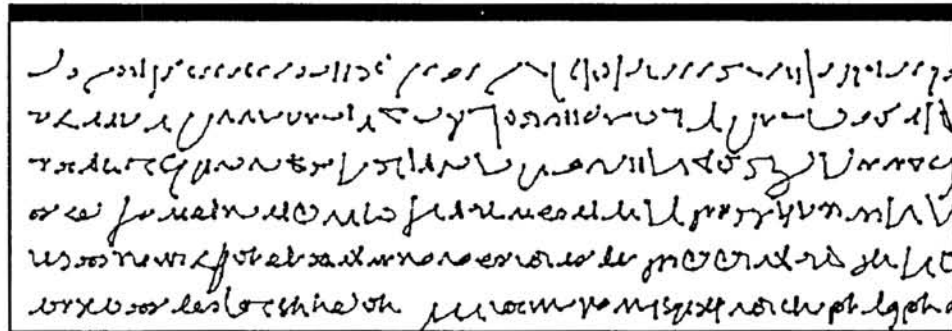

Figure 3: Examples of the class "other" for stroke sizes 1 though 6. Each letter is a random fragment of a word, such that it is not an alphabetic letter.

## 2.2    Letter Recognition

The letter classifier consists of six separate pyramid-style neural-networks, each with an architecture suitable for recognizing a letter of one through six strokes. A neural network designed to recognize letters of size $j$ strokes encodes a mapping from a sequence of $j + 2$ stroke descriptions to a 32 bit error-correcting code [Dietterich & Bakiri, 91]. Experiments have shown this use of a context window improves performance, since the allograph of the current letter is dependent on the allographs of the previous and following letters. The network architecture for stroke size two is illustrated in Figure 2. The architecture is similar to a time-delayed neural-network [Lang & Waibel, 90] in that the hierarchical structure enables different levels of abstract features to be learned. However, the individual receptive fields are *not* shared as in a TDNN, since translational variance is not problem and the sequence of data is important.

The networks are trained using 80% of the raw data collected. This set is further divided into a training and a verification set. All training and verification data is preprocessed and hand segmented, via a graphical interface, into letter samples. These are then sorted according to size and assembled into distinct training and verification sets. It is often the case that the same letter will appear in multiple size files due to variability in writing and different contexts (such as when an $o$ is followed by a $g$ it is at least a 3 stroke allograph, while an $o$ followed by an $l$ is usually only a two stroke allograph). Included in these letter samples are samples of a new letter class "other," illustrated in Figure 3. Experiments demonstrated that use of an "other" class tightens decision boundaries and thus prevents spurious fragments—of which there are many during performance—from being recognized as real letters. Each network is trained using back-propagation until correctness on the verification set is maximized, usually requiring less than 100 epochs.

## 2.3    Word Interpreter

To identify the correct word, the word interpreter explores the space of all possible segmentations of the input stroke sequence. First, the input sequence is partitioned into all possible fragments of size one through six, then the appropriately sized network is used to classify each fragment. An example of this process is illustrated as a matrix in Figure 4(a).

The word interpreter then performs a search of this matrix to identify candidate words. Figure 4(b) and Figure 4(c) illustrates two sets of candidate words found for the example in Figure 4(a). Candidates in this search process are generated according to the following constraints:

- A legal segmentation point of the input stream is one where no two adjacent fragments overlap or leave a gap. To impose this constraint the $i$'th fragment of size $j$ may be extended by all of the $i + j$ fragments, if they exist.

- A legal candidate letter sequence must be a subsequence of a word in the provided lexicon of expected English words.

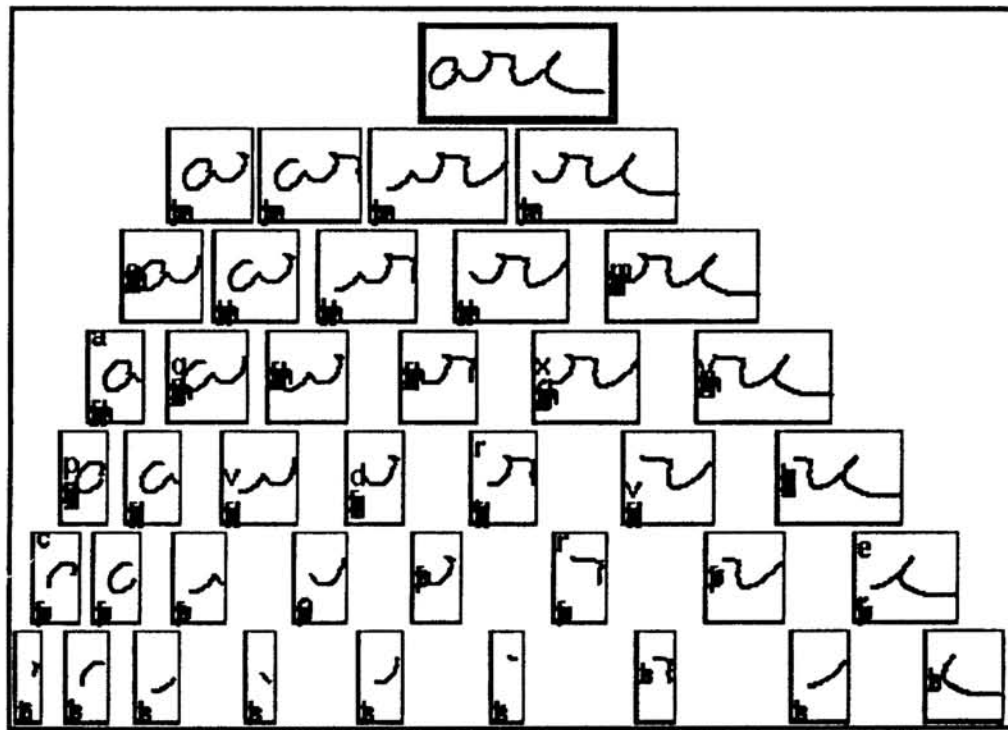

(a)

(b)                                        (c)

Figure 4: (a) The matrix of fragments and their classifications that is generated by
applying the letter recognizers to a sample of the word *are*. The original handwriting
sample, following preprocessing, is given at the top of the matrix. The bottom row
of the matrix corresponds to all fragments of size one (with zero overlap), the second
row to all fragments of size two (with an overlap of one stroke) etc. The column
of letters in each fragment box represents the letter classifications generated by
the neural network of appropriate size. The higher the letter in the column, the
more confident the classification. Those fragments with no high scoring letter were
recognized as examples of the class "other." (b) The first five candidates found by
the word recognizer employing no lexicon. The first column is the word recognized,
the second column is the score for that word, the third is the sequence of fragments
and their classifications. (c) The first five candidates found by the word recognizer
employing a lexicon of 10748 words.

In a forward search, a candidate of size $n$ consists of: (a) a legal sequence of fragments $f_1, f_2, \ldots, f_n$ that form a prefix of the input stroke sequence, (b) a sequence of letters $l_1, l_2, \ldots, l_n$ that form a prefix of an English word from the given dictionary and (c) a score $s$ for this candidate, defined as the average letter recognition error:

$$s = \frac{\sum_{i=1}^{n} \delta(f_i, l_i)}{n}$$

where $\delta(f_i, l_i)$ is the hamming distance between letter $l_i$'s code and the actual code produced by the neural network when given $f_i$ as input. This scoring function is the same as employed in [Edelman *et. al*, 90].

The best word candidate is one that conforms with the constraints and has the lowest score. Although this is a reasonable scoring function, it is easy to show that it is not *admissible* when used as an evaluation function in forward search. With a forward search, problems arise when the prefix of the correct word is poorly recognized. To help combat this problem without greatly increasing the size of the search space, both forward and backward search is performed.

Search is initiated by first generating all one letter and one fragment prefix and suffix candidates. Then at each step in the search, the candidate with the lowest score is expanded by considering the cross product of all legal letter extensions (according to the lexicon) with all legal fragment extensions (according to the fragment-sequence constraints). The list of candidates is maintained as a heap for efficiency. The search process terminates when the best candidate satisfies: (1) the letter sequence is a complete word in the lexicon and (2) the fragment sequence uses all the available input strokes.

The result of this bi-directional search process is illustrated in Figure 4(a)(b), where the five best candidates found are given for no lexicon and a large lexicon. The use of a 10,748 word lexicon eliminates meaningless fragment sequences, such as *cvre*, which is a reasonable segmentation, but not in the lexicon. The first two candidates are the same fragment sequence, found by the two search directions. The third candidate with a 10,748 word dictionary illustrates an alternative segmentation of the correct word. This candidate was identified by a backward search, but not a forward search, due to the poor recognition of the first fragment.

## 3   Evaluation

To evaluate the system, 10 writers have provided samples of approximately 100 words picked by a random process, biased to better represent uncommon letters. Two kinds of experiments were performed. First, to test the ability of the system to learn a variety of writing styles, the system was tested and trained on distinct sets of samples from the *same* writer. This experiment was repeated 10 times, once for each writer. The error rate varied between 1.7% and 15.5%, with a mean of 6.2%, when employing a database of 10,748 English words. The second experiments tested the ability of the system to recognize handwriting of a writer not represented in the training set. Here the set of 10 samples were split into two sets, the training set of 9 writers with the remaining 1 writer being the test set. The error rate was understandably higher, varying between 5.2% and 31.1%, with a mean of 10.8%, when employing a database of 10,748 English words.

## 4  Summary

This paper has presented a recognition-based segmentation approach for on-line cursive handwriting. The method is very flexible because segmentation is performed following exhaustive recognition. Hence, we expect the method to be successful with more natural unconstrained writing, which can include mixed block, cursive and disjoint letters, diverse orderings of delayed strokes, overwrites and erasures.

**Acknowledgements**

This work was supported by a Utah State University Faculty Grant. Thanks to Balaji Allamapatti, Rebecca Rude and Prashanth G Bilagi for code development.

**References**

[Dietterich & Bakiri, 91] Dietterich, T., G. & Bakiri, G. (1991). Error correcting output codes: A general method for improving multiclass inductive learning programs, in *Proceedings of the Ninth National Conference on Artificial Intelligence*, Vol 2, pp 572-577.

[Edelman *et. al*, 90] Edelman S., Tamar F., and Ullman S. (1990). Reading cursive handwriting by alignment of letter prototypes. *International Journal of Computer Vision*, 5:3, 303-331.

[Guerfali & Plamondon, 93] Guerfali W. & Plamondon R. (1993). Normalizing and restoring on-line handwriting. *Pattern Recognition*, Vol. 26, No. 3, pp. 419–431.

[Guyon *et. al*, 90] Guyon I., Albrecht P., Le Cun Y., Denker J. & Hubbard W. (1991). Design of a neural network character recognizer for a touch terminal. *Pattern Recognition*, Vol. 24, No. 2. pp. 105–119.

[Lang & Waibel, 90] Lang K., J. & Waibel A., H. (1990). A time-delayed neural network architecture for isolated word recognition, *Neural Networks*, Vol 3, pp 33-43.

[Morasso *et. al*, 93] Morasso P., Barberis, S. Pagliano S. & Vergano, D. (1993). Recognition experiments of cursive dynamic handwriting with self-organizing networks. *Pattern Recognition*, Vol. 26, No. 3, pp. 451–460.

[Schenkel *et al.*, 93] Schenkel M., Weissman H., Guyon I., Nohl C., & Henderson D. (1993). Recognition-based segmentation of on-line hand-printed words. In S. J. Hanson, J. D. Cowan & C. L. Giles (Eds), *Advances in Neural Information Processing Systems*, 5, 723–730. San Mateo, CA: Morgan Kaufmann.

[Schomaker, 1993] Schomaker L. (1993). Using stroke or character based self-organizing maps in the recognition of on-line connected cursive script. *Pattern Recognition*, Vol. 26. No. 3., pp. 442–450.

[Srihari & Bozinovic, 87] Srihari S. N. & Bozinovic R. M. (1987). A multi-level perception approach to reading cursive script. *Artificial Intelligence*, **33** 217–255.

[Weissman *et. al*, 93] Weissman H., Schenkel M., Guyon I., Nohl C. & Henderson D. (1993). Recognition-based segmentation of on-line run-on hand printed words: input vs. output segmentation. *Pattern Recognition*.